# On fast approximate submodular minimization

**Stefanie Jegelka**[†]**, Hui Lin**[∗]**, Jeff Bilmes**[∗]
[†] Max Planck Institute for Intelligent Systems, Tuebingen, Germany
[∗] University of Washington, Dept. of EE, Seattle, U.S.A.
`jegelka@tuebingen.mgp.de,{hlin,bilmes}@ee.washington.edu`

## Abstract

We are motivated by an application to extract a representative subset of machine learning training data and by the poor empirical performance we observe of the popular minimum norm algorithm. In fact, for our application, minimum norm can have a running time of about $O(n^7)$ ($O(n^5)$ oracle calls). We therefore propose a fast approximate method to minimize arbitrary submodular functions. For a large sub-class of submodular functions, the algorithm is exact. Other submodular functions are iteratively approximated by tight *submodular* upper bounds, and then repeatedly optimized. We show theoretical properties, and empirical results suggest significant speedups over minimum norm while retaining higher accuracies.

## 1 Introduction

Submodularity has been and continues to be an important property in many fields. A set function $f : 2^{\mathcal{V}} \to \mathbb{R}$ defined on subsets of a finite *ground set* $\mathcal{V}$ is submodular if it satisfies the inequality $f(S) + f(T) \geq f(S \cup T) + f(S \cap T)$ for all $S, T \subseteq \mathcal{V}$. Submodular functions include entropy, graph cuts (defined as a function of graph nodes), potentials in many Markov Random Fields [3], clustering objectives [23],covering functions (e.g., sensor placement objectives), and many more. One might consider submodular functions as being on the boundary between "efficiently", i.e., polynomial-time, and "not efficiently" optimizable set functions. Submodularity is gaining importance in machine learning too, but many machine learning data sets are so large that mere "polynomial-time" efficiency is not enough. Indeed, the submodular function minimization (SFM) algorithms with proven polynomial running time are practical only for very small data sets. An alternative, often considered to be faster in practice, is the minimum-norm point algorithm [7]. Its worst-case running time however is still an open question.

Contrary to current wisdom, we demonstrate that for certain functions relevant in practice (see Section 1.1), the minimum-norm algorithm has an impractical empirical running time of about $O(n^7)$, requiring about $O(n^5)$ oracle function calls. To our knowledge, and interesting from an optimization perspective, this is worse than any results reported in the literature, where times of $O(n^{3.3})$ were obtained with simpler graph cut functions [22].

Since we found the minimum-norm algorithm to be either slow (when accurate), or inaccurate (when fast), in this work we take a different approach. We view the SFM problem as an instance of a larger class of problems that *includes* NP-hard instances. This class admits approximation algorithms, and we apply those instead of an exact method. Contrary to the possibly poor performance of "exact" methods, our approximate method is fast, is exact for a large class of submodular functions, and approximates all other functions with bounded deviation.

Our approach combines two ingredients: 1) the representation of functions by graphs; and 2) a recent generalization of graph cuts that combines edge-costs non-linearly. Representing functions as graph cuts is a popular basis for optimization, but cuts cannot efficiently represent all submodular functions. Contrary to previous constructions, including 2) leads to exact representations for *any* submodular

function. To optimize an arbitrary submodular function $f$ represented in our formalism, we construct a graph-representable tractable submodular upper bound $\hat{f}$ that is tight at a given set $T \subseteq \mathcal{V}$, i.e., $\hat{f}(T) = f(T)$, and $\hat{f}(S) \geq f(S)$ for all $S \subseteq \mathcal{V}$. We repeat this "submodular majorization" step and optimize, in at most a linear number of iterations. The resulting algorithm efficiently computes good approximate solutions for our motivating application and other difficult functions as well.

## 1.1 Motivating application and the failure of the minimum-norm point algorithm

Our motivating problem is how to empirically evaluate new or expensive algorithms on large data sets without spending an inordinate amount of time doing so [20, 21]. If a new idea ends up performing poorly, knowing this sooner will avoid futile work. Often the complexity of a training iteration is linear in the number of samples $n$ but *polynomial* in the number $c$ of classes or *types*. For example, for object recognition, it typically takes $O(c^k)$ time to segment an image into regions that each correspond to one of $c$ objects, using an MRF with non-submodular $k$-interaction potential functions. In speech recognition, moreover, a $k$-gram language model with size-$c$ vocabulary has a complexity of $O(c^k)$, where $c$ is in the hundreds of thousands and $k$ can be as large as six.

To reduce complexity one can reduce $k$, but this can be unsatisfactory since the novelty of the algorithm might entail this very cost. An alternative is to extract and use a subset of the training data, one with small $c$. We would want any such subset to possess the richness and intricacy of the original data while simultaneously ensuring that $c$ is bounded.

This problem can be solved via SFM using the following *Bipartite neighborhoods* class of submodular functions: Define a bipartite graph $\mathcal{H} = (\mathcal{V}, \mathcal{U}, \mathcal{E}, w)$ with left/right nodes $\mathcal{V}/\mathcal{U}$, and a modular weight function $w : \mathcal{U} \to \mathbb{R}_+$. A function is *modular* if $w(U) = \sum_{u \in U} w(u)$. Let the neighborhood of a set $S \subseteq \mathcal{V}$ be $\mathcal{N}(S) = \{u \in \mathcal{U} : \exists \text{ edge } (i, u) \in \mathcal{E} \text{ with } i \in S\}$. Then $f : 2^{\mathcal{V}} \to \mathbb{R}_+$, defined as $f(S) = \sum_{u \in \mathcal{N}(S)} w(u)$, is non-decreasing submodular. This function class encompasses e.g. set covers of the form $f(S) = |\bigcup_{i \in S} U_i|$ for sets $U_i$ covered by element $i$. We say $f$ is the submodular function *induced* by modular function $w$ and graph $\mathcal{H}$.

Let $\mathcal{U}$ be the set of types in a set of training samples $\mathcal{V}$. More-over, let $w$ measure the cost of a type $u \in \mathcal{U}$ (this corresponds e.g. to the "undesirability" of type $u$). Define also a modular function $m : 2^{\mathcal{V}} \to \mathbb{R}_+$, $m(S) = \sum_{i \in S} m(i)$ as the benefit of training samples (e.g., in vision, $m(i)$ is the number of different objects in an image $i \in \mathcal{V}$, and in speech, this is the length of utterance $i$). Then the above optimization problem can be solved by finding $\operatorname{argmin}_{S \subseteq \mathcal{V}} w(\mathcal{N}(S)) - \lambda m(S) = \operatorname{argmin}_{S \subseteq \mathcal{V}} w(\mathcal{N}(S)) + \lambda m(\mathcal{V} \setminus S)$ where $\lambda$ is a tradeoff coefficient. As shown below, this can be easily represented and solved efficiently via graph cuts. In some cases, however, we prefer to pick certain subclasses of $\mathcal{U}$ *together*. We partition $\mathcal{U} = \mathcal{U}_1 \cup \mathcal{U}_2$

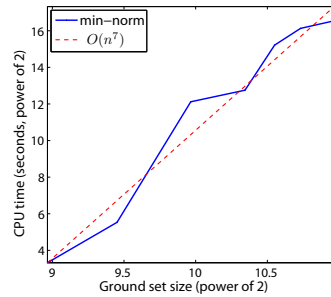

Figure 1: Running time of MN

into blocks, and make it beneficial to pick items from the same block. Benefit restricted to blocks can arise from non-negative non-decreasing submodular functions $g : 2^{\mathcal{U}} \to \mathbb{R}_+$ restricted to blocks. The resulting optimization problem is $\min_{S \subseteq \mathcal{V}} \sum_i g(\mathcal{U}_i \cap \mathcal{N}(S)) + \lambda m(\mathcal{V} \setminus S)$; the sum over $i$ expresses the obvious generalization to a partition into more than just two blocks. Unfortunately, this class of submodular functions is no longer representable by a bipartite graph, and general SFM must be used.

With such a function, $f(S) = m(S) + 100\sqrt{w(\mathcal{N}(S))}$, the empirical running time of the minimum norm point algorithm (MN) scales as $O(n^7)$, with $O(n^5)$ oracle calls (Figure 1). This rules out large data sets for our application, but is interesting with regard to the unknown complexity of MN.

## 1.2 Background on Algorithms for submodular function minimization (SFM)

The first polynomial algorithm for SFM was by Grötschel et al. [13], with further milestones being the first combinatorial algorithms [15, 27] ([22] contains a survey). The currently fastest strongly polynomial combinatorial algorithm has a running time of $O(n^5 T + n^6)$ [24] (where $T$ is function evaluation time), far from practical. Thus, the minimum-norm algorithm [7] is often the method of choice.

Luckily, many sub-families of submodular functions permit specialized, faster algorithms. Graph cut functions fall into this category [1]. They have found numerous applications in computer vision [2, 12], begging the question as to which functions can be represented and minimized using graph cuts [9, 6, 31]. Živný et al. [32] show that cut representations are indeed limited: even when allowing exponentially many additional variables, not all submodular functions can be expressed as graph cuts. Moreover, to maintain efficiency, we do not wish to add too many auxiliary variables, i.e., graph nodes. Other specific cases of relatively efficient SFM include graphic matroids [25] and symmetric submodular functions, minimizable in cubic time [26].

A further class of benign functions are those of the form $f(S) = \psi(\sum_{i \in S} w(i)) + m(S)$ for nonnegative weights $w : \mathcal{V} \to \mathbb{R}_+$, and certain concave functions $\psi : \mathbb{R} \to \mathbb{R}$. Fujishige and Iwata [8] minimize such a function via a parametric max-flow, and we build on their results in Section 4. However, restrictions apply to the effective number of breakpoints of $\psi$. Stobbe and Krause [29] generalize this class to arbitrary concave functions and exploit Nesterov's accelerated gradient descent. Whereas Fujishige and Iwata [8] decompose $\psi$ as a minimum of modular functions, Stobbe and Krause [29] decompose it into a sum of truncated functions of the form $f(A) = \min\{\sum_{i \in A} w'(i), \gamma\}$ — this class of functions, however, is also limited. Truncations are expressible by graph cuts, as we show in Figure 3(b). Thus, if truncations could express any submodular function, then so could graph cuts, contradicting the results in [32]. This was proven independently in [30]. Moreover, the formulation itself of some representable functions in terms of concave functions can be challenging.

In this paper, by contrast, we propose a model that is exact for graph-representable functions, and yields an approximation for all other functions.

## 2 Representing submodular functions by generalized graph cuts

We begin with the representation of a set function $f : 2^{\mathcal{V}} \to \mathbb{R}$ by a graph cut, and then extend this to submodular edge weights. Formally, $f$ is graph-representable if there exists a graph $\mathcal{G} = (\mathcal{V} \cup \mathcal{U} \cup \{s, t\}, \mathcal{E})$ with terminal nodes $s, t$, one node for each element $i$ in $\mathcal{V}$, a set $\mathcal{U}$ of auxiliary nodes ($\mathcal{U}$ can be empty), and edge weights $w : \mathcal{E} \to \mathbb{R}_+$ such that, for any $S \subseteq \mathcal{V}$:

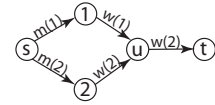

$$f(S) = \min_{U \subseteq \mathcal{U}} w(\delta(s \cup S \cup U)) = \min_{U \subseteq \mathcal{U}} \sum_{e \in \delta_s(S \cup U)} w(e). \quad (1)$$

Figure 2: max

$\delta(S)$ is the set of edges leaving $S$, and $\delta_s(S) = \delta(\{s\} \cup S)$. Recall that any minimal $(s, t)$-cut partitions the graph nodes into the set $T_s \subseteq \mathcal{V} \cup \mathcal{U}$ reachable from $s$ and the set $T_t = (\mathcal{V} \cup \mathcal{U}) \setminus T_s$ disconnected from $s$. That means, $f(S)$ equals the weight of the minimum $(s, t)$-cut that assigns $S$ to $T_s$ and $\mathcal{V} \setminus S$ to $T_t$, and the auxiliary nodes to achieve the minimum. The nodes in $\mathcal{U}$ act as auxiliary variables. As an illustrative example, Figure 2 represents the function $f(S) = \max_{i \in S} w(i) + \sum_{j \in \mathcal{V} \setminus S} m(j)$ for two elements $\mathcal{V} = \{1, 2\}$ and $w(2) > w(1)$, using one auxiliary node $u$. For any query set $S$, $u$ might be joined with $S$ ($u \in T_s$) or not ($u \in T_t$). If $S = \{1\}$, then $w(\delta_s(\{1, u\})) = m(2) + w(2)$, and $w(\delta_s(\{1\})) = m(2) + w(1) = f(S) < w(\delta_s(\{1, u\}))$. If $S = \{1, 2\}$, then $w(\delta_s(\{1, 2, u\})) = w(2) < w(\delta_s(\{1, 2\})) = w(1) + w(2)$, and indeed $f(S) = w(2)$. The graph representation (1) leads to the equivalence between minimum cuts and the minimizers of $f$:

**Lemma 1.** *Let $S^*$ be a minimizer of $f$, and let $U^* \in \operatorname{argmin}_{U \subseteq \mathcal{U}} w(\delta_s(S^* \cup U))$. Then the boundary $\delta_s(S^* \cup U^*) \subseteq \mathcal{E}$ is a minimum cut in $\mathcal{G}$.*

The lemma (proven in [18]) is good news since minimum cuts can be computed efficiently. To derive $S^*$ from a minimum cut, recall that any minimum cut is the boundary of some set $T_s^* \subseteq \mathcal{V} \cup \mathcal{U}$ that is still reachable from $s$ after cutting. Then $S^* = T_s^* \cap \mathcal{V}$, so $S^* \subseteq T_s^*$ and $(\mathcal{V} \setminus S^*) \subseteq T_t^*$. A large sub-family of submodular functions can be expressed exactly in the form (1), but possibly with an exponentially large $\mathcal{U}$. For efficiency, the size of $\mathcal{U}$ should remain small. To express *any* submodular function with *few* auxiliary nodes, in this paper we extend Equation (1) as is seen below.

Unless the submodular function $f$ is already a graph cut function (and directly representable), we first decompose $f$ into a modular function and a nondecreasing submodular function, and then build up the graph part by part. This accounts for any graph-representable component of $f$. To approximate the remaining component of the function that is not exactly representable, we use submodular costs on **graph edges** (in contrast with graph nodes), a construction that has been introduced recently in

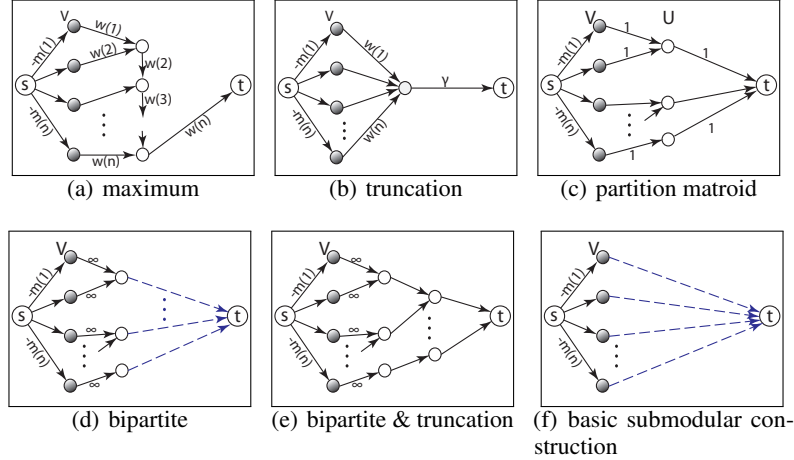

(a) maximum             (b) truncation         (c) partition matroid

(d) bipartite    (e) bipartite & truncation   (f) basic submodular construction

Figure 3: Example graph constructions. Dashed blue edges can have submodular weights; auxiliary nodes are white and ground set nodes are shaded. The bipartite graph can have arbitrary representations between $\mathcal{U}$ and $t$, 3(e) is one example. (All figures are best viewed in color.)

computer vision [16]. We first introduce a relevant decomposition result by Cunningham [4]. A polymatroid rank function is *totally normalized* if $f(\mathcal{V} \setminus i) = f(\mathcal{V})$ for all $i \in \mathcal{V}$. The marginal costs are defined as $\rho_f(i|S) = f(S \cup \{i\}) - f(S)$ for all $i \in \mathcal{V} \setminus S$.

**Theorem 1** ([4, Thm. 18]). *Any submodular function $f$ can be decomposed as $f(S) = m(S) + g(S)$ into a modular function $m$ and a totally normalized polymatroid rank function $g$. The components are defined as $m(S) = \sum_{i \in A} \rho_f(i|\mathcal{V} \setminus i)$ and $g(S) = f(S) - m(S)$ for all $S \subseteq \mathcal{V}$.*

We may assume that $m(i) < 0$ for all $i \in \mathcal{V}$. If $m(i) \geq 0$ for any $i \in \mathcal{V}$, then diminishing marginal costs, a property of submodular functions, imply that we can discard element $i$ immediately [5, 18]. To express such negative costs in a graph cut, we point out an equivalent formulation with positive weights: since $m(\mathcal{V})$ is constant, minimizing $m(S) = \sum_{i \in S} m(i)$ is equivalent to minimizing the shifted function $m(S) - m(\mathcal{V}) = -m(\mathcal{V} \setminus S)$. Thus, we instead minimize the sum of positive weights on the complement of the solution. We implement this shifted function in the graph by adding an edge $(s, i)$ with *nonnegative* weight $-m(i)$ for each $i \in \mathcal{V}$. Every element $j \in T_t$ (i.e., $j \notin S$) that is not selected must be separated from $s$, and the edge $(s, j)$ contributes $-m(j)$ to the total cut cost.

Having constructed the modular part of the function $f$ by edges $(s, i)$ for all $i \in \mathcal{V}$, we address its submodular part $g$. If $g$ is a sum of functions, we can add a subgraph for each function. We begin with some example functions that are explicitly graph-representable with polynomially many auxiliary nodes $\mathcal{U}$. The illustrations in Figure 3 include the modular part $m$ as well.

**Maximum.** The function $g(S) = \max_{i \in S} w(i)$ for nonnegative weights $w$ is an extension of Figure 2. Without loss of generality, we assume the elements to be ordered by weight, so that $w(1) \leq w(2) \leq \ldots w(n)$. We introduce $n-1$ auxiliary nodes $u_j$, and connect them to form an imbalanced tree with leaves $\mathcal{V}$, as illustrated in Figure 3(a). The minimum way to disconnect a set $S$ from $t$ is to cut the single edge $(u_{j-1}, u_j)$ with weight $w(j)$ of the largest element $j = \operatorname{argmax}_{i \in S} w(i)$.

**Truncations.** Truncated functions $f(S) = \min\{w(S), \gamma\}$ for $w, \gamma \geq 0$ can be modeled by one extra variable, as shown in Figure 3(b). If $w(S) > \gamma$, then the minimization in (1) puts $u$ in $T_s$ and cuts the $\gamma$-edge. This construction has been successfully used in computer vision [19]. Truncations can model piecewise linear concave functions of $w(S)$ [19, 29], and also represent negative terms in a pseudo-boolean polynomial [18]. Furthermore, these functions include rank functions $g(S) = \min\{|S|, k\}$ of uniform matroids, and rank functions of partition matroids. If $\mathcal{V}$ is partitioned into groups $G \subset \mathcal{V}$, then the rank of the associated partition matroid counts the number of groups that $S$ intersects: $f(S) = |\{G|G \cap S \neq \emptyset\}|$ (Fig. 3(c)).

**Bipartite neighborhoods.** We already encountered bipartite submodular functions $f(S) = \sum_{u \in \mathcal{N}(S)} w(u)$ in Section 1.1. The bipartite graph that defines $\mathcal{N}(S)$ is part of the representa-

tion shown in Figure 3(d), and its edges get infinite weight. As a result, if $S \in T_s$, then all neighbors $\mathcal{N}(S)$ of $S$ must also be in $T_s$, and the edges $(u, t)$ for all $u \in \mathcal{N}(S)$ are cut. Each $u \in \mathcal{U}$ has such an edge $(u, t)$, and the weight of that edge is the weight $w(u)$ of $u$.

Additional examples are given in [18].

Of course, all the above constructions can also be applied to subsets $Q \subset \mathcal{V}$ of nodes. In fact, the decomposition and constructions above permit us to address arbitrary sums and restrictions of such graph-representable functions. These example families of functions already cover a wide variety of functions needed in applications. Minimizing a graph-represented function is equivalent to finding the minimum $(s, t)$-cut, and all edge weights in the above are nonnegative. Thus we can use any efficient min-cut or max-flow algorithm for any of the above functions.

## 2.1 Submodular edge weights

Next we address the generic case of a submodular function that is not (efficiently) graph-representable or whose functional form is unknown. We can still decompose this function into a modular part $m$ and a polymatroid $g$. Then we construct a simple graph as shown in Figure 3(f). The representation of $m$ is the same as above, but the cost of the edges $(i, t)$ will be charged differently. Instead of a sum of weights, we define the cost of a set of these edges to be a non-additive function on sets of edges, a polymatroid rank function. Each edge $(i, t)$ is associated with exactly one ground set element $i \in \mathcal{V}$, and selecting $i$ ($i \in T_s$) is equivalent to cutting the edge $(i, t)$. Thus, the cost of edge $(i, t)$ will model the cost $g(i)$ of its element $i \in \mathcal{V}$. Let $\mathcal{E}_t$ be the set of such edges $(i, t)$, and denote, for any subset $C \subseteq \mathcal{E}_t$ the set of ground set elements adjacent to $C$ by $V(C) = \{i \in \mathcal{V} | (i, t) \in C\}$. Equivalently, $C$ is the boundary of $V(C)$ in $\mathcal{E}_t$: $\delta_s(V(C)) \cap \mathcal{E}_t = C$. We define the cost of $C$ to be the cost of its adjacent ground set elements, $h_g(C) \triangleq g(V(C))$; this implies $h_g(\delta_s(S \cap \mathcal{E}_t)) = g(S)$. The equivalent of Equation (1) becomes

$$f(S) = \min_{U \subseteq \mathcal{U}} w(\delta_s(S \cup U) \setminus \mathcal{E}_t) + h_g(\delta_s(S \cup U) \cap \mathcal{E}_t) = -m(\mathcal{V} \setminus S) + g(S), \qquad (2)$$

with $\mathcal{U} = \emptyset$ in Figure 3(f). This generalization from the standard sum of edge weights to a nondecreasing submodular function permits us to express many more functions, in fact *any* submodular function [5]. Such expressiveness comes at a price, however: *in general*, finding a minimum $(s, t)$-cut with such submodular edge weights is NP-hard, and even hard to approximate [17]. The graphs here that represent *submodular* functions correspond to benign examples that are not NP-hard. Nevertheless, we will use an approximation algorithm that applies to all such non-additive cuts. We describe the algorithm in Section 3. For the moment, we assume that we can handle submodular costs on edges.

The simple construction in Figure 3(f) itself corresponds to a general submodular function minimization. It becomes powerful when combined with parts of $f$ that are explicitly representable. If $g$ decomposes into a sum of graph-representable functions and a (nondecreasing submodular) remainder $g_r$, then we construct a subgraph for each graph-representable function, and combine these subgraphs with the submodular-edge construction for $g_r$. All the subgraphs share the same ground set nodes $\mathcal{V}$. In addition, we are in no way restricted to separating graph-representable and general submodular functions. The cost function in our application is a submodular function *induced* by a bipartite graph $\mathcal{H} = (\mathcal{V}, \mathcal{U}, \mathcal{E})$. Let, as before, $\mathcal{N}(S)$ be the neighborhood of $S \subseteq \mathcal{V}$ in $\mathcal{U}$. Given a nondecreasing submodular function $g_{\mathcal{U}} : 2^{\mathcal{U}} \to \mathbb{R}_+$ on $\mathcal{U}$, the graph $\mathcal{H}$ defines a function $g(S) = g_{\mathcal{U}}(\mathcal{N}(S))$. If $g_{\mathcal{U}}$ is nondecreasing submodular, then so is $g$ [28, §44.6 g]. For any such function, we represent $\mathcal{H}$ explicitly in $\mathcal{G}$, and then add submodular-cost edges from $\mathcal{U}$ to $t$ with $h_g(\delta_s(\mathcal{N}(S))) = g_{\mathcal{U}}(\mathcal{N}(S))$, as shown in Figure 3(d). If $g_{\mathcal{U}}$ is itself exactly representable, then we add the appropriate subgraph instead (Figure 3(e)).

# 3 Optimization

To minimize a function $f$, we find a minimum $(s, t)$-cut in its representation graph. Algorithm 1 applies to any submodular-weight cut; this algorithm is exact if the edge costs are modular (a sum of weights). In each iteration, we approximate $f$ by a function $\hat{f}$ that is efficiently graph-representable, and minimize $\hat{f}$ instead. In this section, we switch from costs $f, \hat{f}$ of node sets $S, T$ to equivalent costs $w, h$ of edge sets $A, B, C$ and back.

**Algorithm 1**: Minimizing graph-based approximations.

create the representation graph $\mathcal{G} = (\mathcal{V} \cup \mathcal{U} \cup \{s, t\}, \mathcal{E})$ and set $S_0 = T_0 = \emptyset$;
**for** $i = 1, 2, \ldots$ **do**
    compute edge weights $\nu_{i-1} = \nu_{\delta_s(T_{i-1})}$ (Equation 4);
    find the (maximal) minimum $(s, t)$-cut $T_i = \operatorname{argmin}_{T \subseteq (\mathcal{V} \cup \mathcal{U})} \nu_{i-1}(\delta_s T)$;
    **if** $f(T_i) = f(T_{i-1})$ **then**
        return $S_i = T_i \cap \mathcal{V}$;
    **end**
**end**

The approximation $\hat{f}$ arises from the cut representation constructed in Section 2: we replace the exact edge costs by approximate modular edge weights $\nu$ in $\mathcal{G}$. Recall that the representation $\mathcal{G}$ has two types of edges: those whose weights $w$ are counted as the usual sum, and those charged via a submodular function $h_g$ derived from $g$. We denote the latter set by $\mathcal{E}_t$, and the former by $\mathcal{E}_m$. For any $e \in \mathcal{E}_m$, we use the exact cost $\nu(e) = w(e)$. The submodular cost $h_g$ of the remaining edges is upper bounded by referring to a fixed set $B \subseteq \mathcal{E}$ that we specify later. For any $A \subseteq \mathcal{E}_t$, we define

$$\hat{h}_B(A) \triangleq h_g(B) + \sum_{e \in A \setminus B} \rho_h(e | B \cap \mathcal{E}_t) - \sum_{e \in B \setminus A} \rho_h(e | \mathcal{E}_t \setminus e) \geq h_g(A). \tag{3}$$

This inequality holds thanks to diminishing marginal costs, and the approximation is tight at $B$, $\hat{h}_B(B) = h_g(B)$. Up to a constant shift, this function is equivalent [16] to the edge weights:

$$\nu_B(e) = \rho_h(e | B \cap \mathcal{E}_t) \quad \text{if } e \in \mathcal{E}_t \setminus B; \qquad \text{and} \qquad \nu_B(e) = \rho_h(e | \mathcal{E}_t \setminus e) \quad \text{if } e \in B \cap \mathcal{E}_t. \tag{4}$$

Plugging $\nu_B$ into Equation (2) yields an approximation $\hat{f}$ of $f$. In the algorithm, $B$ is always the boundary $B = \delta_s(T)$ of a set $T \subseteq (\mathcal{V} \cup \mathcal{U})$. Then $\mathcal{G}$ with weights $\nu_B$ represents

$$\hat{f}(S) = \min_{U \subseteq \mathcal{U}} \ \nu_B(\delta_s(S \cup U) \cap \mathcal{E}_m) + \nu_B(\delta_s(S \cup U) \cap \mathcal{E}_t)$$

$$= \min_{U \subseteq \mathcal{U}} \ w(\delta_s(S \cup U) \cap \mathcal{E}_m) + \sum_{(u,t) \in \delta_s(S \cup U) \cap B} \rho_g(u | \mathcal{V} \cup \mathcal{U} \setminus u) + \sum_{(u,t) \in \delta_s(S \cup U) \setminus B} \rho_g(u | T).$$

Here, we used the definition $h_g(C) \triangleq g(V(C))$. Importantly, the edge weights $\nu_B$ are always nonnegative, because, by Theorem 1, $g$ is guaranteed to be nondecreasing. Hence, we can efficiently minimize $\hat{f}$ as a standard minimum cut. If in Algorithm 1 there is more than one set $T$ defining a minimum cut, then we pick the largest (i.e., maximal) such set. Lemma 2 states properties of the $T_i$.

**Lemma 2.** *Assume $\mathcal{G}$ is any of the graphs in Figure 3, and let $T^* \subseteq \mathcal{V} \cup \mathcal{U}$ be the maximal set defining a minimum-cost cut $\delta_s(T^*)$ in $\mathcal{G}$, so that $S^* = T^* \cap \mathcal{V}$ is a minimizer of the function represented by $\mathcal{G}$. Then, in any iteration $i$ of Algorithm 1, it holds that $T_{i-1} \subseteq T_i \subseteq T^*$. In particular, $S \subseteq S^*$ for the returned solution $S$.*

Lemma 2 has three important implications. First, the algorithm never picks any element outside the maximal optimal solution. Second, because the $T_i$ are growing, there are at most $|T^*| \leq |\mathcal{V} \cup \mathcal{U}|$ iterations, and the algorithm is strongly polynomial. Finally, the chain property permits more efficient implementations. The proof of Lemma 2 relies on the definition of $\nu$ and submodularity [18]. Moreover, the weights $\nu$ lead to a bound the worst-case approximation factor [18].

### 3.1 Improvement via summarizations

The approximation $\hat{f}$ is loosest if the sum of edge weights $\nu_i(A)$ significantly overestimates the true joint cost $h_g(A)$ of sets of edges $A \subseteq \delta_s T^* \setminus \delta_s T_i$ still to be cut. This happens if the joint marginal cost $\rho_h(A | \delta_s T_i)$ is much smaller than the estimated sum of weights, $\nu_i(A) = \sum_{e \in A} \rho_h(e | \delta_s T_i)$. Luckily, many of the functions that show this behavior strongly resemble truncations. Thus, to tighten the approximation, we summarize the joint cost of groups of edges by a construction similar to Figure 3(b). Then the algorithm can take larger steps and pick groups of elements.

We partition $\mathcal{E}_t$ into disjoint groups $G_k$ of edges $(u, t)$. For each group, we introduce an auxiliary node $t_k$ and re-connect all edges $(u, t) \in G_k$ to end in $t_k$ instead of $t$. Their cost remains the

same. An extra edge $e_k$ connects $t_k$ to $t$, and carries the joint weight $\nu_i(e_k)$ of all edges in $G_k$; a tighter approximation. The weight $\nu_i(e_k)$ is also adapted in each iteration. Initially, we set $\nu_0(e_k) = h_g(G_k) = g(V(G_k))$. Subsequent approximations $\nu_i$ refer to cuts $\delta_s T_i$, and such a cut can contain either single edges from $G_k$, or the group edge $e_k$. We set the next reference set $B_i$ to be a copy of $\delta_s T_i$ in which each group edge $e_k$ was replaced by all its group members $G_k$. The joint group weight $\nu_i(e_k)$ for any $k$ is then $\nu_i(e_k) = \rho_h(G_k \setminus B_i | B_i) + \sum_{e \in G_k \cap B_i} \rho_h(e | \mathcal{E}_t \setminus e) \le \sum_{e \in G_k} \nu_i(e)$. Formally, these weights represent the upper bound

$$\hat{h}'_B(A) = h_g(B) + \sum_{G_k \subseteq A} \rho_h(G_k \setminus B | B) + \sum_{e \in (G_k \cap A) \setminus B, G_k \not\subseteq A} \rho_h(e | B) - \sum_{e \in B \setminus A} \rho_h(e | \mathcal{E}_t \setminus e) \le \hat{h}(A),$$

where we replace $G_k$ by $e_k$ whenever $G_k \subseteq A$. In our experiments, this summarization helps improve the results while simultaneously reducing running time.

## 4 Parametric constructions for special cases

For certain functions of the form $f(S) = m(S) + g(\mathcal{N}(S))$, the graph representation in Figure 3(d) admits a specific algorithm. We use approximations that are exact on limited ranges, and eventually pick the best range. For this construction, $g$ must have the form $g(U) = \psi(\sum_{u \in U} \tilde{w}(u))$ for weights $\tilde{w} \ge 0$ and one piecewise linear, concave function $\psi$ with a small (polynomial) number $\ell$ of breakpoints. Alternatively, $\psi$ can be *any* concave function if the weights $\tilde{w}$ are such that $\tilde{w}(U) = \sum_{u \in U} \tilde{w}(u)$ can take at most polynomially many distinct values $x_k$; e.g., if $\tilde{w}(u) = 1$ for all $u$, then effectively $\ell = |\mathcal{U}| + 1$ by using the $x_k$ as breakpoints and interpolating. In all these cases, $\psi$ is equivalent to the minimum of at most $\ell$ linear (modular) functions.

We build on the approach in [8], but, whereas their functions are defined on $\mathcal{V}$, $g$ here is defined on $\mathcal{U}$. Contrary to their functions and owing to our decomposition, the $\psi$ here is nondecreasing. We define $\ell$ linear functions, one for each breakpoint $x_k$ (and use $x_0 = 0$):

$$\psi_k(t) = (\psi(x_k) - \psi(x_{k-1}))(t - x_k) + \psi(x_k) = \alpha_k t + \beta_k. \tag{5}$$

The $\psi_k$ are defined such that $\psi(t) = \min_k \psi_k(t)$. Therefore, we approximate $f$ by a series $\hat{f}_k(S) = -m(\mathcal{V} \setminus S) + \psi_k(\tilde{w}(\mathcal{N}(S)))$, and find the exact minimizer $S_k$ for each $k$. To compute $S_k$ via a minimum cut in $\mathcal{G}$ (Fig. 3(d)), we define edge weights $\nu_k(e) = w(e)$ for edges $e \notin \mathcal{E}_t$ as in Section 3, and $\nu_k(u, t) = \alpha_k \tilde{w}(u)$ for $e \in \mathcal{E}_t$. Then $T_k = S_k \cup \mathcal{N}(S_k)$ defines a minimum cut $\delta_s T_k$ in $\mathcal{G}$. We compute $\hat{f}_k(S_k) = \nu_k(\delta_s T_k) + \beta_k + m(\mathcal{V})$; the optimal solution is the $S_k$ with minimum cost $\hat{f}_k(S_k)$. This method is exact. To solve for all $k$ within one max-flow, we use a parametric max-flow method [10, 14]. Parametric max-flow usually works with both edges from $s$ and to $t$. Here, $\nu_k \ge 0$ because $\psi$ is nondecreasing, and thus we only need $t$-edges which already exist in the bipartite graph $\mathcal{G}$.

This method is limited to few breakpoints. For more general concave $\psi$ and arbitrary $\tilde{w} \ge 0$, we can approximate $\psi$ by a piecewise linear function. Still, the parametric approach does not directly generalize to more than one nonlinearity, e.g., $g(U) = \sum_i g_i(U \cap W_i)$ for sets $W_i \subseteq \mathcal{U}$. In contrast, Algorithm 1 (with the summarization) can handle all of these cases. We point out that without indirection via the bipartite graph, i.e., $f(S) = m(S) + \psi(w(S))$ for a $\psi$ with few breakpoints, we can minimize $f$ very simply: The solution for $\psi_k$ includes all $j \in \mathcal{V}$ with $\alpha_k \le -m(j)/w(j)$. The advantage of the graph cut is that it easily combines with other objectives.

## 5 Experiments

In the experiments, we test whether the graph-based methods improve over the minimum-norm point algorithm in the difficult cases of Section 1.1. We compare the following methods:
**MN:** a re-implementation of the minimum norm point algorithm in C++ that is about four times faster than the C code in [7] (see [18]), ensuring that our results are not due to a slow implementation;
**MC:** a minimum cut with static edge weights $\nu(e) = h_g(e)$;
**GI:** the graph-based iterative Algorithm 1, implemented in C++ with the max-flow code of [3], (i) by itself; (ii) with summarization via $\sqrt{|\mathcal{E}_t|}$ random groups (GIr); (iii) with summarization via groups generated by sorting the edges in $\mathcal{E}_t$ by their weights $h_g(e)$, and then forming groups $G_k$ of edges adjacent in the order such that for each $e \in G_k$, $h_g(e) \le 1.1 h_g(G_k)$ (GIs);

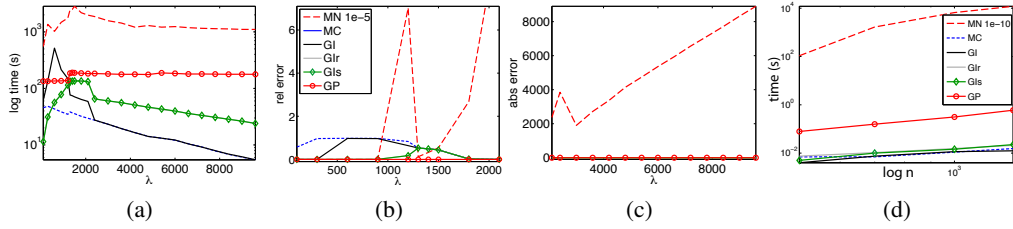

Figure 4: (a) Running time, (b) relative and (c) absolute error with varying $\lambda$ for a data set as described in Section 1.1, $|\mathcal{V}| = 54915$, $|\mathcal{U}| = 6871$, and $f(S) = -m(S) + \lambda\sqrt{|\mathcal{N}(S)|}$. Where $f(S^*) = 0$, we show absolute errors. (d) Running times with respect to $|\mathcal{V}|$, $f(S) = -m(S) + \lambda\sqrt{w(\mathcal{N}(S))}$.

**GP:** the parametric method from Section 4, using $|\mathcal{E}_t|$ equispaced breakpoints; based on C code from RIOT[1].

We also implemented the SLG method from [29] in C++ (public code is not available), but found it to be impractical on the problems here, as gradient computation of our function requires finding gradients of $|\mathcal{U}|$ truncation functions, which is quite expensive [18]. Thus, we did not include it in the tests on the large graphs. We use bipartite graphs of the form described in Section 1.1, with a cost function $f(S) = m(S) + \lambda g(\mathcal{N}(S))$. The function $g$ uses a square root, $g(U) = \sqrt{w(U)}$. More results, also on other functions, can be found in [18].

**Solution quality with solution size.** Running time and results depend on the size of $S^*$. Thus, we vary $\lambda$ from 50 ($S^* \approx \mathcal{V}$) to 9600 ($S^* = \emptyset$) on a speech recognition data set [11]. The bipartite graph represents a corpus subset extraction problem (Section 1.1) and has $|\mathcal{V}| = 54915$, $|\mathcal{U}| = 6871$ nodes, and uniform weights $w(u) = 1$ for all $u \in U$. The results look similar with non-uniform weights, but for uniform weights the parametric method from Section 4 always finds the optimal solution and thus allows us to report errors. Figure 4 shows the running times and the relative error $\text{err}(S) = |f(S) - f(S^*)|/|f(S^*)|$ (note that $f(S^*) \leq 0$). If $f(S^*) = 0$, we report absolute errors. Because of the large graph, we used the minimum-norm algorithm with accuracy $10^{-5}$. Still, it takes up to 100 times longer than the other methods. It works well if $S^*$ is large, but as $\lambda$ grows, its accuracy becomes poor. In particular when $f(S^*) = f(\emptyset) = 0$, it returns large sets with large positive cost. In contrast, the deviation of the approximate edge weights $\nu_i$ from the true cost is bounded [18]. All algorithms except MN return an optimal solution for $\lambda \geq 2000$. Updating the weights $\nu$ clearly improves the performance of Algorithm 1, as does the summarization (GIr/GIs perform identically here). With the latter, the solutions are very often optimal, and almost always very good.

**Scaling:** To test how the methods scale with the size $|\mathcal{V}|$, we sample small graphs from the big graph, and report average running times across 20 graphs for each size. As the graphs have non-uniform weights, we use GP as an approximation method and estimate the nonlinearity $\sqrt{w(U)}$ by a piecewise linear function with $|\mathcal{U}|$ breakpoints. All algorithms find the same (optimal) solution. Figure 4(d) shows that the minimum-norm algorithm with high accuracy is much slower than the other methods. Empirically, MN scales as up to $O(n^5)$ (note that Figure 1 is a *specific* worst-case graph), the parametric version approximately $O(n^2)$, and the variants of GI up to $O(n^{1.5})$.

**Acknowledgments:** This material is based upon work supported in part by the National Science Foundation under grant IIS-0535100, by an Intel research award, a Microsoft research award, and a Google research award.

# References

[1] R. K. Ahuja, T. L. Magnanti, and J. B. Orlin. *Network Flows*. Prentice Hall, 1993.

[2] Y. Boykov and M.-P. Jolly. Interactive graph cuts for optimal boundary and region segmentation of objects in n-d images. In *ICCV*, 2001.

---

[1]http://riot.ieor.berkeley.edu/riot/Applications/Pseudoflow/parametric.html

[3] Y. Boykov and V. Kolmogorov. An experimental comparison of min-cut/max-flow algorithms for energy minimization in vision. *IEEE TPAMI*, 26(9):1124–1137, 2004.

[4] W. H. Cunningham. Decomposition of submodular functions. *Combinatorica*, 3(1):53–68, 1983.

[5] W. H. Cunningham. Testing membership in matroid polyhedra. *J Combinatorial Theory B*, 36:161–188, 1984.

[6] D. Freedman and P. Drineas. Energy minimization via graph cuts: Settling what is possible. In *CVPR*, 2005.

[7] S. Fujishige and S. Isotani. A submodular function minimization algorithm based on the minimum-norm base. *Pacific Journal of Optimization*, 7:3–17, 2011.

[8] S. Fujishige and S. Iwata. Minimizing a submodular function arising from a concave function. *Discrete Applied Mathematics*, 92, 1999.

[9] S. Fujishige and S. B. Patkar. Realization of set functions as cut functions of graphs and hypergraphs. *Discrete Mathematics*, 226:199–210, 2001.

[10] G. Gallo, M.D. Grigoriadis, and R.E. Tarjan. A fast parametric maximum flow algorithm and applications. *SIAM J Computing*, 18(1), 1989.

[11] J.J. Godfrey, E.C. Holliman, and J. McDaniel. Switchboard: Telephone speech corpus for research and development. In *Proc. ICASSP*, volume 1, pages 517–520, 1992.

[12] D. M. Greig, B. T. Porteous, and A. H. Seheult. Exact maximum a posteriori estimation for binary images. *Journal of the Royal Statistical Society*, 51(2), 1989.

[13] M. Grötschel, L. Lovász, and A. Schrijver. The ellipsoid algorithm and its consequences in combinatorial optimization. *Combinatorica*, 1:499–513, 1981.

[14] D. Hochbaum. The pseudoflow algorithm: a new algorithm for the maximum flow problem. *Operations Research*, 58(4), 2008.

[15] S. Iwata, L. Fleischer, and S. Fujishige. A combinatorial strongly polynomial algorithm for minimizing submodular functions. *J. ACM*, 48:761–777, 2001.

[16] S. Jegelka and J. Bilmes. Submodularity beyond submodular energies: coupling edges in graph cuts. In *CVPR*, 2011.

[17] S. Jegelka and J. Bilmes. Approximation bounds for inference using cooperative cuts. In *ICML*, 2011.

[18] S. Jegelka, H. Lin, and J. Bilmes. Fast approximate submodular minimization: Extended version, 2011.

[19] P. Kohli, L. Ladický, and P. Torr. Robust higher order potentials for enforcing label consistency. *Int. J. Computer Vision*, 82, 2009.

[20] H. Lin and J. Bilmes. An application of the submodular principal partition to training data subset selection. In *NIPS workshop on Discrete Optimization in Machine Learning*, 2010.

[21] H. Lin and J. Bilmes. Optimal selection of limited vocabulary speech corpora. In *Proc. Interspeech*, 2011.

[22] S. T. McCormick. Submodular function minimization. In K. Aardal, G. Nemhauser, and R. Weismantel, editors, *Handbook on Discrete Optimization*, pages 321–391. Elsevier, 2006. updated version 3a (2008).

[23] M. Narasimhan, N. Jojic, and J. Bilmes. Q-clustering. In *NIPS*, 2005.

[24] J. B. Orlin. A faster strongly polynomial time algorithm for submodular function minimization. *Mathematical Programming*, 118(2):237–251, 2009.

[25] M. Preissmann and A. Sebő. *Research Trends in Combinatorial Optimization*, chapter Graphic Submodular Function Minimization: A Graphic Approach and Applications, pages 365–385. Springer, 2009.

[26] M. Queyranne. Minimizing symmetric submodular functions. *Mathematical Programming*, 82:3–12, 1998.

[27] A. Schrijver. A combinatorial algorithm minimizing submodular functions in strongly polynomial time. *J. Combin. Theory Ser. B*, 80:346–355, 2000.

[28] A. Schrijver. *Combinatorial Optimization*. Springer, 2004.

[29] P. Stobbe and A. Krause. Efficient minimization of decomposable submodular functions. In *NIPS*, 2010.

[30] J. Vondrák. personal communication, 2011.

[31] S. Živný and P.G. Jeavons. Classes of submodular constraints expressible by graph cuts. *Constraints*, 15: 430–452, 2010. ISSN 1383-7133.

[32] S. Živný, D. A. Cohen, and P. G. Jeavons. The expressive power of binary submodular functions. *Discrete Applied Mathematics*, 157(15):3347–3358, 2009.

